# The effect of correlated input data on the dynamics of learning

**Søren Halkjær and Ole Winther**

CONNECT, The Niels Bohr Institute
Blegdamsvej 17
2100 Copenhagen, Denmark
`halkjaer,winther@connect.nbi.dk`

## Abstract

The convergence properties of the gradient descent algorithm in the case of the linear perceptron may be obtained from the response function. We derive a general expression for the response function and apply it to the case of data with simple input correlations. It is found that correlations severely may slow down learning. This explains the success of PCA as a method for reducing training time. Motivated by this finding we furthermore propose to transform the input data by removing the mean across input variables as well as examples to decrease correlations. Numerical findings for a medical classification problem are in fine agreement with the theoretical results.

## 1 INTRODUCTION

Learning and generalization are important areas of research within the field of neural networks. Although good generalization is the ultimate goal in feed-forward networks (perceptrons), it is of practical importance to understand the mechanism which control the amount of time required for learning, i. e. the dynamics of learning. This is of course particularly important in the case of a large data set. An exact analysis of this mechanism is possible for the linear perceptron and as usual it is hoped that the results to some extend may be carried over to explain the behaviour of non-linear perceptrons.

We consider $N$ dimensional input vectors $\mathbf{x} \in \mathcal{R}^N$ and scalar output $y$. The linear

perceptron is parametrized by the weight vector $\mathbf{w} \in \mathcal{R}^N$

$$y(\mathbf{x}) = \frac{1}{\sqrt{N}}\mathbf{w}^T\mathbf{x} \tag{1}$$

Let the training set be $\{(\mathbf{x}^\mu, y^\mu), \mu = 1, \ldots, p\}$ and the training error be the usual squared error, $E(\mathbf{w}) = \frac{1}{2}\sum_\mu (y^\mu - y(\mathbf{x}^\mu))^2$. We will use the well-known gradient descent algorithm[1] $\mathbf{w}(k+1) = \mathbf{w}(k) - \eta \nabla E(\mathbf{w}(k))$ to estimate the minimum points $\mathbf{w}^*$ of $E$. Here $\eta$ denotes the learning parameter. Collecting the input examples in the $N \times p$ matrix $\mathbf{X}$ and the corresponding output in $\mathbf{y}$, the error function is written $E(\mathbf{w}) = \frac{1}{2}(\mathbf{w}^T\mathbf{R}\mathbf{w} - 2\mathbf{q}^T\mathbf{w} + c)$, where $\mathbf{R} \equiv \frac{1}{N}\sum_\mu \mathbf{x}^\mu(\mathbf{x}^\mu)^T$, $\mathbf{q} = \frac{1}{\sqrt{N}}\mathbf{X}\mathbf{y}$ and $c = \mathbf{y}^T\mathbf{y}$. As in (Le Cun *et al.*, 1991) the convergence properties of the minimum points $\mathbf{w}^*$ are examined in the coordinate system where $\mathbf{R}$ is diagonal. Let $\mathbf{U}$ denote the matrix whose columns are the eigenvectors of $\mathbf{R}$ and $\Delta = \text{diag}(\lambda_1, \ldots, \lambda_N)$ the diagonal matrix containing the eigenvalues of $\mathbf{R}$. The new coordinates then become $\mathbf{v} = \mathbf{U}^T(\mathbf{w} - \mathbf{w}^*)$ with corresponding error function[2]

$$E(\mathbf{v}) = \frac{1}{2}\mathbf{v}^T\Delta\mathbf{v} + E_0 = \frac{1}{2}\sum_i \lambda_i v_i^2 + E_0 \tag{2}$$

where $E_0 = E(\mathbf{w}^*)$. Gradient descent now leads to the decoupled equations

$$v_i(k+1) = (1 - \eta\lambda_i)v_i(k) = (1 - \eta\lambda_i)^k v_i(0) \tag{3}$$

with $i = 1, \ldots, N$. Clearly, $\mathbf{v} \to \mathbf{0}$ requires $|1 - \eta\lambda_i| < 1$ for all $i$, so that $\eta$ must be chosen in the interval $0 < \eta < 2/\lambda_{max}$. In the extreme case $\lambda_i = \lambda$ we will have convergence in one step for $\eta = 1/\lambda$. However, in the usual case of unequal $\lambda_i$ the convergence for large $k$ will be exponential $v_i(k) = \exp(-\eta\lambda_i k)v_i(0)$. $(\eta\lambda_i)^{-1}$ therefore defines the time constant of the $i$'th equation giving a slowest time constant $(\eta\lambda_{min})^{-1}$. A popular choice for the learning parameter is $\eta = 1/\lambda_{max}$ resulting in a slowest time constant $\lambda_{max}/\lambda_{min}$ called the learning time $\tau$ in the following. The convergence properties of the gradient descent algorithm is thus characterized by $\tau$. In the case of a singular matrix $\mathbf{R}$, one or more of the eigenvalues will be zero, and there will be no convergence along the corresponding eigendirections. This has however no influence on the error according to (2). Thus, $\lambda_{min}$ will in the following denote the smallest non-zero eigenvalue.

We will in the article calculate the eigenvalue spectrum of $\mathbf{R}$ in order to obtain the learning time of the gradient descent algorithm. This may be done by introducing the response function

$$G_{\mathbf{L}} \equiv G(\mathbf{L}, \mathbf{H}) = \frac{1}{N}\text{Tr}\mathbf{L}\frac{1}{1 - \mathbf{R}\mathbf{H}} \equiv \left\langle \mathbf{L}\frac{1}{1 - \mathbf{R}\mathbf{H}} \right\rangle \tag{4}$$

where $\mathbf{L}, \mathbf{H}$ are arbitrary $N \times N$ matrices. Using a standard representation of the Dirac $\delta$-function (Krogh, 1992) we may write the eigenvalue spectrum of $\mathbf{R}$ as

$$\rho(\lambda) = \frac{1}{N}\sum_i \delta(\lambda - \lambda_i) = -\frac{1}{\pi}\text{Im}\,G(\lambda^{-1}, \lambda^{-1}) \tag{5}$$

where $\lambda$ has an infinitesimal imaginary part which is set equal to zero at the end of the calculation.

In the 'thermodynamic limit' $N \rightarrow \infty$ keeping $\alpha = \frac{p}{N}$ constant and finite, $G$ (and thus the eigenvalue spectrum) is a self-averaging quantity (Sollich, 1996) i. e. $G - \overline{G} = \mathcal{O}(N^{-1})$, where $\overline{G}$ is defined as the response function averaged over the input distribution. Previously $\overline{G}$ has been calculated for independent input variables (Hertz *et al.*, 1989; Sollich, 1996). In section 2 we derive an implicit equation for the averaged response function for arbitrary correlations using random matrix techniques (Brody *et al.*, 1981). This equation is solved showing that simple input correlations may slow down learning significantly. Based on this finding we propose in section 3 data transformations for improving the learning speed and test the transformation numerically on a medical classification problem in section 4. We conclude in section 5 with a discussion of the results.

## 2  THE RESPONSE FUNCTION

The method for deriving the averaged response function is based on the fact that the response function (4) may be written as a geometrical series $G_{\mathbf{L}} = \sum_{r=0}^{\infty} \langle \mathbf{L}(\mathbf{R}\mathbf{H})^r \rangle$. We will assume that the input examples $\mathbf{x}^\mu$ are drawn independently from a Gaussian distribution with means $m_i$ and correlations $\overline{x_i x_j} - m_i m_j = C_{ij}$, i. e. $\overline{\mathbf{x}^\mu (\mathbf{x}^\nu)^T} = \delta_{\mu\nu} \mathbf{Z}$ and $\overline{\mathbf{R}} = \alpha \mathbf{Z}$ where $\mathbf{Z} \equiv \mathbf{C} + \mathbf{m}\mathbf{m}^T$. The Gaussian distribution has the property that the average of products of $x$'s can be calculated by making all possible pair correlations, e.g. $\overline{x_i x_j x_k x_l} = Z_{ij} Z_{kl} + Z_{ik} Z_{jl} + Z_{il} Z_{jk}$. To take the average of $\langle \mathbf{L}(\mathbf{R}\mathbf{H})^r \rangle$, we must therefore make all possible pairs of the $\mathbf{x}$'s and exchange each pair $x_i x_j$ with $Z_{ij}$. This combinatorial problem will be solved below in a recursive fashion leading to an implicit equation for $\overline{G}_L$. Using underbraces to indicate pairings of $\mathbf{x}$'s, we get for $r \geq 2$

$$
\begin{aligned}
\overline{\langle \mathbf{L}(\mathbf{R}\mathbf{H})^r \rangle} &= \frac{1}{N} \sum_\mu \overline{\left\langle \mathbf{L} \underbrace{\mathbf{x}^\mu (\mathbf{x}^\mu)^T} \mathbf{H}(\mathbf{R}\mathbf{H})^{r-1} \right\rangle} \\
&+ \frac{1}{N^2} \sum_{s=0}^{r-2} \sum_{\mu,\nu} \overline{\left\langle \mathbf{L}\mathbf{x}^\mu \underbrace{(\mathbf{x}^\mu)^T \mathbf{H}(\mathbf{R}\mathbf{H})^s \mathbf{x}^\nu} (\mathbf{x}^\nu)^T \mathbf{H}(\mathbf{R}\mathbf{H})^{r-s-2} \right\rangle} \\
&= \alpha \overline{\langle \mathbf{L}\mathbf{Z}\mathbf{H}(\mathbf{R}\mathbf{H})^{r-1} \rangle} + \sum_{s=0}^{r-2} \overline{\langle \mathbf{L}(\mathbf{R}\mathbf{H})^{r-s-1} \rangle} \; \overline{\langle \mathbf{Z}\mathbf{H}(\mathbf{R}\mathbf{H})^s \rangle}
\end{aligned} \tag{6}
$$

Resumming this we get the response function

$$
\overline{G}_{\mathbf{L}} = \langle \mathbf{L} \rangle + \alpha \sum_{r=0}^{\infty} \overline{\langle \mathbf{L}\mathbf{Z}\mathbf{H}(\mathbf{R}\mathbf{H})^r \rangle} + \sum_{r=0}^{\infty} \sum_{s=0}^{r-2} \overline{\langle \mathbf{L}(\mathbf{R}\mathbf{H})^{r-s-1} \rangle} \; \overline{\langle \mathbf{Z}\mathbf{H}(\mathbf{R}\mathbf{H})^s \rangle} \tag{7}
$$

Exchanging the order of summation in the last term we can write everything in terms of the response function

$$
\begin{aligned}
\overline{G}_{\mathbf{L}} &= \langle \mathbf{L} \rangle + \alpha \overline{G}_{\mathbf{LZH}} + \sum_{s=0}^{\infty} \sum_{r=s+1}^{\infty} \overline{\langle \mathbf{L}(\mathbf{R}\mathbf{H})^{r-s} \rangle} \; \overline{\langle \mathbf{Z}\mathbf{H}(\mathbf{R}\mathbf{H})^s \rangle} \\
&= \langle \mathbf{L} \rangle + \alpha \overline{G}_{\mathbf{LZH}} + (\overline{G}_{\mathbf{L}} - \langle \mathbf{L} \rangle) \overline{G}_{\mathbf{CH}}
\end{aligned}
$$

$$= \langle \mathbf{L} \rangle + \frac{\alpha \overline{G}_{\text{LZH}}}{1 - \overline{G}_{\text{ZH}}} \tag{8}$$

Using (8) recursively setting $\mathbf{L}$ equal to $\mathbf{LZH}$, $\mathbf{L(ZH)}^2$ etc. one obtains

$$\overline{G}_{\mathbf{L}} = \sum_{r=0}^{\infty} \left\langle \mathbf{L} \left( \frac{\alpha \mathbf{ZH}}{1 - \overline{G}_{\text{ZH}}} \right)^r \right\rangle = \left\langle \mathbf{L} \frac{1}{1 - \frac{\alpha \mathbf{ZH}}{1 - \overline{G}_{\text{ZH}}}} \right\rangle \tag{9}$$

This is our main result for the response function. To get the response function $\overline{G}_{\frac{1}{\lambda}} = G(\lambda^{-1}, \lambda^{-1})$ requires two steps, first set $\mathbf{L} = \mathbf{ZH}$ and solve for $\overline{G}_{\text{ZH}}$ and then solve for $\overline{G}_{\frac{1}{\lambda}}$. If $\mathbf{Z}$ has a particularly simple form (9) may be solved analytically, but in general it must be solved numerically.

In the following we will calculate the eigenvalue spectrum for a correlation matrix on the form $\mathbf{C} = \mathbf{nn}^T + r\mathbf{I}$ and general mean $\mathbf{m}$. To ensure that $\mathbf{C}$ is positive semi definite $r \geq 0$ and $|\mathbf{n}|^2 + r \geq 0$ where $|\mathbf{n}|^2 \equiv \mathbf{n} \cdot \mathbf{n}$. The eigenvalues of $\mathbf{Z} = \mathbf{nn}^T + \mathbf{mm}^T + r\mathbf{I}$ are straight forwardly shown to be $a_1 = r$ (with multiplicity $N - 2$), $a_2 = r + \left[ |\mathbf{n}|^2 + |\mathbf{m}|^2 - \sqrt{D} \right]/2$ and $a_3 = r + \left[ |\mathbf{n}|^2 + |\mathbf{m}|^2 + \sqrt{D} \right]/2$ with $D = (|\mathbf{n}|^2 - |\mathbf{m}|^2)^2 + 4(\mathbf{n} \cdot \mathbf{m})^2$. Carrying out the trace in eq. (9) we get

$$\overline{G}_{\frac{1}{\lambda}} = \frac{N-2}{N} \frac{1}{\lambda - \frac{\alpha a_1}{1 - \overline{G}_{\text{ZH}}}} + \frac{1}{N} \frac{1}{\lambda - \frac{\alpha a_2}{1 - \overline{G}_{\text{ZH}}}} + \frac{1}{N} \frac{1}{\lambda - \frac{\alpha a_3}{1 - \overline{G}_{\text{ZH}}}} \tag{10}$$

This expression suggests that we may solve $\overline{G}_{\frac{1}{\lambda}}$ in powers of $1/N$ (see e.g. (Sollich, 1996)). However for purposes of the discussion of learning times the only $\frac{1}{N}$-term that will be of importance is the last term above. We therefore only need to solve for $\overline{G}_{\text{ZH}}$ (setting $\mathbf{L} = \mathbf{ZH}$ in (9)) to leading order

$$\overline{G}_{\text{ZH}} = \frac{\lambda + a_1(1-\alpha) - \sqrt{(\lambda + a_1(1-\alpha)^2) - 4\lambda a_1}}{2\lambda} \tag{11}$$

Note that $\overline{G}_{\text{ZH}}$ will vanish for large $\lambda$ implying that the last term in (10) to leading order is singular for $\lambda = \alpha a_3$. Inserting the result in (10) gives

$$\overline{G}_{\frac{1}{\lambda}} = \frac{1}{2\lambda a_2} \left[ \lambda + a_2(1-\alpha) - \sqrt{(\lambda + a_2(1-\alpha))^2 - 4\lambda a_2} \right] + \frac{1}{N} \frac{1}{\lambda - \alpha a_1} \tag{12}$$

According to (5) the eigenvalue spectrum is determined by the imaginary part and poles of $\overline{G}_{\frac{1}{\lambda}}$. $\overline{G}_{\frac{1}{\lambda}}$ has an imaginary part for $\lambda_- < \lambda < \lambda_+$ where $\lambda_\pm = a_1(1 \pm \sqrt{\alpha})^2$ and the poles $\lambda = 0$, $\lambda = \alpha a_3$. The poles contribute each with a $\delta$-function such that the eigenvalue spectrum up to corrections of order $\frac{1}{N}$ becomes

$$\overline{\rho}(\lambda) = (1-\alpha)\Theta(1-\alpha)\delta(\lambda) + \frac{1}{N}\delta(\lambda - \alpha a_3) + \frac{1}{2\pi\lambda a_1}\sqrt{(\lambda_+ - \lambda)(\lambda - \lambda_-)} \tag{13}$$

where $\Theta(x) = 1$ for $x > 0$ and 0 otherwise. The first term expresses the trivial fact that for $p < N$ the whole input space is not spanned and $\mathbf{R}$ will have a fraction of $1 - \alpha$ zero-eigenvalues. The continuous spectrum (the root term) only contributes for $\lambda_- < \lambda < \lambda_+$. Numerical simulations has been performed to test the validity of the spectrum (13) (Halkjær, 1996). They are in good agreement with predicted results indicating that finite size effects are unimportant. The continuous spectrum ·˦ (13) has also been calculated using the replica method (Halkjær, 1996).

From the spectrum the learning time $\tau$ may be read of directly

$$\tau = \max\left(\frac{\lambda_+}{\lambda_-}, \frac{\alpha a_3}{\lambda_-}\right) = \max\left(\left(\frac{1+\sqrt{\alpha}}{1-\sqrt{\alpha}}\right)^2, \frac{\alpha a_3}{a_1\left(1-\sqrt{\alpha}\right)^2}\right) \tag{14}$$

To illustrate how input correlations and bias may affect learning time consider simple correlations $C_{ij} = \delta_{ij}v(1-c) + vc$ and $m_i = m$. With this special choice $\tau = \frac{\alpha N(m^2+vc)}{v(1-c)\left(1-\sqrt{\alpha}\right)^2}$. For $m^2 + cv > 0$, i.e. for non-zero mean or positive correlations, the convergence time will blow up by a factor proportional to $N$. The input bias effect has previously been observed by (Le Cun *et al.*,Wendemuth *et al.*). In the next section we will consider transformations to remove the large eigenvalue and thus to speed up learning.

## 3  DATA TRANSFORMATIONS FOR INCREASING LEARNING SPEED

In this section we consider two data transformations for minimizing the learning time $\tau$ of a data set, based on the results obtained in the previous sections.

The PCA transformation (Jackson, 1991) is a data transformation often used in data analysis. Let $\mathbf{U}$ be the matrix whose columns are the eigenvectors of the sample covariance matrix and let $\mathbf{x}^{mean}$ denote the sample average vector (see below). It is easy to check that the transformed data set

$$\tilde{\mathbf{x}}^\mu = \mathbf{U}^T(\mathbf{x}^\mu - \mathbf{x}^{mean}) \tag{15}$$

have uncorrelated (zero-mean) variables. However, the new PCA variables will often have a large spread in variances which might result in slow convergence. A simple rescaling of the new variables will remove this problem, such that according to (14) a PCA transformed data set with rescaled variables will have optimal convergence properties.

The other transformation, which we will call *double centering*, is based on the removal of the observation means and the variable means. However, whereas the PCA transformation doesn't care about the initial distribution, this transformation is optimal for a data set generated from the matrix $Z_{ij} = \delta_{ij}v(1-c)+vc+m_im_j$ studied earlier. Define $x_i^{mean} = \frac{1}{p}\sum_\mu x_i^\mu$ (mean of the $i$'th variable), $x_{mean}^\mu = \frac{1}{N}\sum_i x_i^\mu$ (mean of the $\mu$'th example) and $x_{mean}^{mean} = \frac{1}{pN}\sum_{\mu i} x_i^\mu$ (grand mean). Consider first the transformed data set

$$\tilde{x}_i^\mu = x_i - x_i^{mean} - x_{mean}^\mu + x_{mean}^{mean}$$

The new variables are readily seen to have zero mean, variance $\tilde{v} = v(1-c) - \frac{v}{N}(1-c)$ and correlation $\tilde{c} = \frac{-1}{N-1}$. Since $\tilde{v}(1-\tilde{c}) = v(1-c)$ we immediately get from (13) that the continuous eigenvalue spectrum is unchanged by this transformation. Furthermore the 'large' eigenvalue $\alpha a_1$ is equal to zero and therefore uninteresting. Thus the learning time becomes $\tau = (1+\sqrt{\alpha})^2/(1-\sqrt{\alpha})^2$. This transformation however removes perhaps important information from the data set, namely the observation means. Motivated by these findings, we create a new data set $\{\tilde{\mathbf{x}}^\mu\}$ where this information is added as an extra component

$$\tilde{\mathbf{x}}^\mu = (\tilde{x}_1^\mu, \ldots, \tilde{x}_N^\mu, x_{mean}^\mu - x_{mean}^{mean}) \tag{16}$$

Table 1: Required number of iterations and corresponding learning times for different data transformations. 'Raw' is the original data set, 'Var. cent.' indicates the variable centered ($m_i = 0$) data set, 'Doub. cent.' denotes (16), while the two last columns concerns the PCA transformed data set (15) without and with rescaled variables.

|  | Raw | Var. cent. | Doub. cent. | PCA | PCA (res.) |
|---|---|---|---|---|---|
| Iterations | $\infty$ | 300 | 50 | 630 | 7 |
| $\tau$ | 161190 | 3330 | 237 | 3330 | 1 |

The matrix $\tilde{\mathbf{R}}$ resulting from this data set is identical to the above case except that a column and a row have been added. We therefore conclude that the eigenvalue spectrum of this data set consists of a continuous spectrum equal to the above and a single eigenvalue which is found to be $\lambda = \frac{1}{N}v(1-c) + cv$. For $c \neq 0$ we will therefore have a learning time $\tau$ of order one indicating fast convergence. For independent variables ($c = 0$) the transformation results in a learning time of order $N$ but in this case a simple removal of the variable means will be optimal. After training, when an $(N+1)$-dim parameter set $\tilde{\mathbf{w}}$ has been obtained, it is possible to transform back to the original data set using the parameter transformation $w_l = \tilde{w}_l + \frac{1}{N}\tilde{w}_{N+1} - \frac{1}{N}\sum_{i=1}^{N}\tilde{w}_i$.

## 4 NUMERICAL INVESTIGATIONS

The suggested transformations for improving the convergence properties have been tested on a medical classification problem. The data set consisted of 40 regional values of cerebral glucose metabolism from 85 patients, 48 HIV-negatives and 37 HIV-positives. A simple perceptron with sigmoidal tanh output was trained using gradient descent on the entropic error function to diagnose the 85 patients correctly. The choice of an entropic error function was due to it's superior convergence properties compared to the quadratic error function considered in the analysis. The learning was stopped once the perceptron was able to diagnose all patients correctly. Table 1 shows the average number of required iterations for each of the transformed data sets (see legend) as well as the ratio $\tau = \lambda_{max}/\lambda_{min}$ for the corresponding matrix $\mathbf{R}$. The 'raw' data set could not be learned within the allowed 1000 iterations which is indicated by an $\infty$. Overall, there's fine agreement between the order of calculated learning times and the corresponding order of required number of iterations. Note especially the superiority of the PCA transformation with rescaled variables.

## 5 CONCLUSION

For linear networks the convergence properties of the gradient descent algorithm may be derived from the eigenvalue spectrum of the covariance matrix of the input data. The convergence time is controlled by the ratio between the largest and smallest (non-zero) eigenvalue. In this paper we have calculated the eigenvalue spectrum of a covariance matrix for correlated and biased inputs. It turns out that correlation and bias give rise to an eigenvalue of order the input dimension as well as a continuous spectrum of order one. This explains why a PCA transformation (with

a variable rescaling) may increase learning speed significantly. We have proposed to center (setting equal to zero) the empirical mean both for each variable and each observation in order to remove the large eigenvalue. We add an additional component containing the observation mean to the input vector in order have this information in the training set. At the end of training it is possible to transform the solution back to the original representation. Numerical investigations are in fine agreement with the theoretical analysis for improving the convergence properties.

## 6 ACKNOWLEDGMENTS

We would like to thank Sara A. Solla and Lars Kai Hansen for valuable comments and discussions. Furthermore we wish to thank Ido Kanter for providing us with notes on some of his previous work. This work has been supported by the Danish National Councils for the Natural and Technical Sciences through the Danish Computational Neural Network Center CONNECT.

**REFERENCES**

Brody, T. A., Flores J., French J. B., Mello, P. A., Pendey, A., & Wong, S. S. (1981) Random-matrix physics. *Rev. Mod. Phys.* 53:385.

Halkjær, S. (1996) *Dynamics of learning in neural networks: application to the diagnosis of HIV and Alzheimer patients.* Master's thesis, University of Copenhagen.

Hertz, J. A., Krogh, A. & Thorbergsson G. I. (1989) Phase transitions in simple learning. *J. Phys. A* 22:2133-2150.

Jackson, J. E. (1991) *A User's Guide to Principal Components.* John Wiley & Sons.

Krogh, A. (1992) Learning with noise in a linear perceptron *J. Phys A* 25:1119-1133.

Le Cun, Y., Kanter, I. & Solla, S.A. (1991) Second Order Properties of Error Surfaces : Learning Time and Generalization. *NIPS*, 3:918-924.

Sollich, P. (1996) Learning in large linear perceptrons and why the thermodynamic limit is relevant to the real world. *NIPS*, 7:207-214

Wendemuth, A., Opper, M. & Kinzel W. (1993) The effect of correlations in neural networks, *J. Phys. A* 26:3165.

## Footnotes

[1]The Newton-Raphson method, $\mathbf{w}(k+1) = \mathbf{w}(k) - \nabla E(\mathbf{w}(k))(\nabla^2 E(\mathbf{w}(k)))^{-1}$ is of course much more effective in the linear case since it gives convergence in one step. This method however requires an inversion of the Hessian matrix.

[2]Note that this analysis is valid for any part of an error surface in which a quadratic approximation is valid. In the general case $\mathbf{R}$ should be exchanged with the Hessian $\nabla\nabla E(\mathbf{w}^*)$.
